# Gradient Weights help Nonparametric Regressors

**Samory Kpotufe***
Max Planck Institute for Intelligent Systems
samory@tuebingen.mpg.de

**Abdeslam Boularias**
Max Planck Institute for Intelligent Systems
boularias@tuebingen.mpg.de

## Abstract

In regression problems over $\mathbb{R}^d$, the unknown function $f$ often varies more in some coordinates than in others. We show that weighting each coordinate $i$ with the estimated norm of the $i$th derivative of $f$ is an efficient way to significantly improve the performance of distance-based regressors, e.g. kernel and $k$-NN regressors. We propose a simple estimator of these derivative norms and prove its consistency. Moreover, the proposed estimator is efficiently learned online.

## 1   Introduction

In regression problems over $\mathbb{R}^d$, the unknown function $f$ might vary more in some coordinates than in others, even though all coordinates might be relevant. How much $f$ varies with coordinate $i$ can be captured by the norm $\|f_i'\|_{1,\mu} = \mathbb{E}_X |f_i'(X)|$ of the $i$th derivative $f_i' = e_i^\top \nabla f$ of $f$. A simple way to take advantage of the information in $\|f_i'\|_{1,\mu}$ is to weight each coordinate proportionally to an estimate of $\|f_i'\|_{1,\mu}$. The intuition, detailed in Section 2, is that the resulting data space behaves as a low-dimensional projection to coordinates with large norm $\|f_i'\|_{1,\mu}$, while maintaining information about all coordinates. We show that such weighting can be learned efficiently, both in batch-mode and online, and can significantly improve the performance of distance-based regressors in real-world applications. In this paper we focus on the distance-based methods of kernel and $k$-NN regression.

For distance-based methods, the weights can be incorporated into a distance function of the form $\rho(x, x') = \sqrt{(x - x')^\top \mathbf{W}(x - x')}$, where each element $\mathbf{W}_i$ of the diagonal matrix $\mathbf{W}$ is an estimate of $\|f_i'\|_{1,\mu}$. This is not *metric learning* [1, 2, 3, 4] where the best $\rho$ is found by optimizing over a sufficiently large space of possible metrics. Clearly metric learning can only yield better performance, but the optimization over a larger space will result in heavier preprocessing time, often $O(n^2)$ on datasets of size $n$. Yet, preprocessing time is especially important in many modern applications where both training and prediction are done online (e.g. robotics, finance, advertisement, recommendation systems). Here we do not optimize over a space of metrics, but rather estimate a *single* metric $\rho$ based on the norms $\|f_i'\|_{1,\mu}$. Our metric $\rho$ is efficiently obtained, can be estimated online, and still significantly improves the performance of distance-based regressors.

To estimate $\|f_i'\|_{1,\mu}$, one does not need to estimate $f_i'$ well everywhere, just well on average. While many elaborate derivative estimators exist (see e.g. [5]), we have to keep in mind our need for fast but consistent estimator of $\|f_i'\|_{1,\mu}$. We propose a simple estimator $\mathbf{W}_i$ which averages the differences along $i$ of an estimator $f_{n,h}$ of $f$. More precisely (see Section 3) $\mathbf{W}_i$ has the form $\mathbb{E}_n |f_{n,h}(X + te_i) - f_{n,h}(X - te_i)| / 2t$ where $\mathbb{E}_n$ denotes the empirical expectation over a sample $\{X_i\}_1^n$. $\mathbf{W}_i$ can therefore be updated online at the cost of just two estimates of $f_{n,h}$.

In this paper $f_{n,h}$ is a kernel estimator, although any regression method might be used in estimating $\|f_i'\|_{1,\mu}$. We prove in Section 4 that, under mild conditions, $\mathbf{W}_i$ is a consistent estimator of the

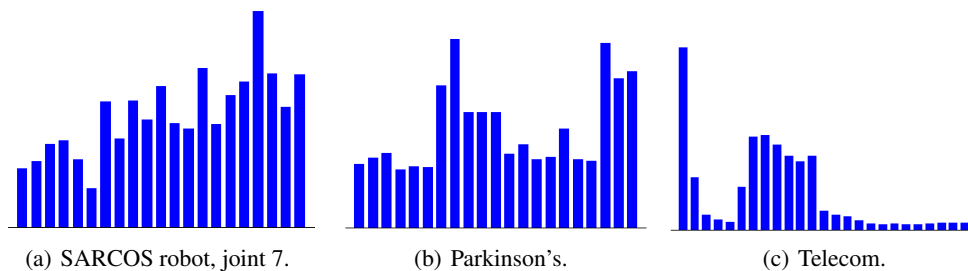

(a) SARCOS robot, joint 7.   (b) Parkinson's.   (c) Telecom.

Figure 1: Typical gradient weights $\left\{ \mathbf{W}_i \approx \|f'_i\|_{1,\mu} \right\}_{i \in [d]}$ for some real-world datasets.

unknown norm $\|f'_i\|_{1,\mu}$. Moreover we prove finite sample convergence bounds to help guide the practical tuning of the two parameters $t$ and $h$.

**Most related work**

As we mentioned above, metric learning is closest in spirit to the gradient-weighting approach presented here, but our approach is different from metric learning in that we do not search a space of possible metrics, but rather estimate a single metric based on gradients. This is far more time-efficient and can be implemented in online applications which require fast preprocessing.

There exists many metric learning approaches, mostly for classification and few for regression (e.g. [1, 2]). The approaches of [1, 2] for regression are meant for batch learning. Moreover [1] is limited to Gaussian-kernel regression, and [2] is tuned to the particular problem of age estimation. For the problem of classification, the metric-learning approaches of [3, 4] are meant for online applications, but cannot be used in regression.

In the case of kernel regression and local polynomial regression, multiple bandwidths can be used, one for each coordinate [6]. However, tuning $d$ bandwidth parameters requires searching a $d \times d$ grid, which is impractical even in batch mode. The method of [6] alleviates this problem, however only in the particular case of local linear regression. Our method applies to any distance-based regressor.

Finally, the ideas presented here are related to recent notions of nonparametric sparsity where it is assumed that the target function is well approximated by a *sparse* function, i.e. one which varies little in most coordinates (e.g. [6], [? ]). Here we do not need sparsity, instead we only need the target function to vary in some coordinates more than in others. Our approach therefore works even in cases where the target function is far from sparse.

## 2 Technical motivation

In this section, we motivate the approach by considering the ideal situation where $\mathbf{W}_i = \|f'_i\|_{1,\mu}$. Let's consider regression on $(\mathcal{X}, \rho)$, where the input space $\mathcal{X} \subset \mathbb{R}^d$ is connected. The prediction performance of a distance-based estimator (e.g. kernel or $k$-NN) is well known to be the sum of its variance and its bias [7]. Regression on $(\mathcal{X}, \rho)$ decreases variance while keeping the bias controlled.

*Regression variance decreases on $(\mathcal{X}, \rho)$:* The variance of a distance based estimate $f_n(x)$ is inversely proportional to the number of samples (and hence the mass) in a neighborhood of $x$ (see e.g. [8]). Let's therefore compare the masses of $\rho$-balls and Euclidean balls. Suppose some weights largely dominate others, for instance in $\mathbb{R}^2$, let $\|f'_2\|_{1,\mu} \gg \|f'_1\|_{1,\mu}$. A ball $B_\rho$ in $(\mathcal{X}, \rho)$ then takes the ellipsoidal shape below which we contrast with the dotted Euclidean ball inside.

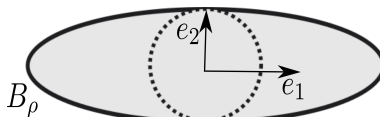

Relative to a Euclidean ball, a ball $B_\rho$ of *similar*[1] radius has more mass in the direction $e_1$ in which $f$ varies least. This intuition is made more precise in Lemma 1 below, which is proved in the appendix. Essentially, let $R \subset [d]$ be the set of coordinates with larger weights $\mathbf{W}_i$, then the mass of balls $B_\rho$ behaves like the mass of balls in $\mathbb{R}^{|R|}$. Thus, effectively, regression in $(\mathcal{X}, \rho)$ has variance nearly as small as that for regression in the lower-dimensional space $\mathbb{R}^{|R|}$.

Note that the assumptions on the marginal $\mu$ in the lemma statement are verified for instance when $\mu$ has a continuous lower-bounded density on $\mathcal{X}$. For simplicity we let $(\mathcal{X}, \|\cdot\|)$ have diameter 1.

**Lemma 1** (Mass of $\rho$-balls). *Consider any $R \subset [d]$ such that $\max_{i \notin R} \mathbf{W}_i < \min_{i \in R} \mathbf{W}_i$. Suppose $\mathcal{X} \equiv \frac{1}{\sqrt{d}}[0,1]^d$, and the marginal $\mu$ satisfies on $(\mathcal{X}, \|\cdot\|)$, for some $C_1, C_2$: $\forall x \in \mathcal{X}, \forall r > 0$, $C_1 r^d \leq \mu(B(x,r)) \leq C_2 r^d$. Let $\kappa \triangleq \sqrt{\max_{i \in R} \mathbf{W}_i / \min_{i \in R} \mathbf{W}_i}$, $\epsilon_{\mathcal{R}} \triangleq \max_{i \notin R} \mathbf{W}_i \cdot \sqrt{d}$, and let $\rho(\mathcal{X}) \triangleq \sup_{x,x' \in \mathcal{X}} \rho(x,x')$.*

*Then for any $\epsilon\rho(\mathcal{X}) > 2\epsilon_{\mathcal{R}}$, $\mu(B_\rho(x, \epsilon\rho(\mathcal{X}))) \geq C(2\kappa)^{-|R|}\epsilon^{|R|}$, where $C$ is independent of $\epsilon$.*

Ideally we would want $|R| \ll d$ and $\epsilon_{\mathcal{R}} \approx 0$, which corresponds to a sparse metric.

*Regression bias remains bounded on $(\mathcal{X}, \rho)$*: The bias of distance-based regressors is controlled by the smoothness of the unknown function $f$ on $(\mathcal{X}, \rho)$, i.e. how much $f$ might differ for two close points. Turning back to our earlier example in $\mathbb{R}^2$, some points $x'$ that were originally far from $x$ along $e_1$ might now be included in the estimate $f_n(x)$ on $(\mathcal{X}, \rho)$. Intuitively, this should not add bias to the estimate since $f$ does not vary much in $e_1$. We have the following lemma.

**Lemma 2** (Change in Lipschitz smoothness for $f$). *Suppose each derivative $f'_i$ is bounded on $\mathcal{X}$ by $|f'_i|_{sup}$. Assume $\mathbf{W}_i > 0$ whenever $|f'_i|_{sup} > 0$. Denote by $R$ the largest subset of $[d]$ such that $|f'_i|_{sup} > 0$ for $i \in R$. We have for all $x, x' \in X$,*

$$|f(x) - f(x')| \leq \left( \sum_{i \in R} \frac{|f'_i|_{sup}}{\sqrt{\mathbf{W}_i}} \right) \rho(x, x').$$

Applying the above lemma with $\mathbf{W}_i = 1$, we see that in the original Euclidean space, the variation in $f$ relative to distance between points $x, x'$, is of the order $\sum_{i \in R} |f'_i|_{sup}$. This variation in $f$ is now increased in $(\mathcal{X}, \rho)$ by a factor of $1/\inf_{i \in R} \sqrt{\|f'_i\|_{1,\mu}}$ in the worst case. In this sense, the space $(\mathcal{X}, \rho)$ maintains information about all relevant coordinates. In contrast, information is lost under a projection of the data in the likely scenario that all or most coordinates are relevant.

Finally, note that if all weights were close, the space $(\mathcal{X}, \rho)$ is essentially equivalent to the original $(\mathcal{X}, \|\cdot\|)$, and we likely neither gain nor loose in performance, as confirmed by experiments. However, we observed that in practice, even when all coordinates are relevant, the gradient-weights vary sufficiently (Figure 1) to observe significant performance gains for distance-based regressors.

## 3 Estimating $\|f'_i\|_{1,\mu}$

In all that follows we are given $n$ i.i.d samples $(\mathbf{X}, \mathbf{Y}) = \{(X_i, Y_i)\}_{i=1}^n$, from some unknown distribution with marginal $\mu$. The marginal $\mu$ has support $\mathcal{X} \subset \mathbb{R}^d$ while the output $Y \in \mathbb{R}$.

The kernel estimate at $x$ is defined using any kernel $K(u)$, positive on $[0, 1/2]$, and 0 for $u > 1$. If $B(x,h) \cap \mathbf{X} = \emptyset$, $f_{n,h}(x) = \mathbb{E}_n Y$, otherwise

$$f_{n,\bar{\rho},h}(x) = \sum_{i=1}^n \frac{K(\bar{\rho}(x, X_i)/h)}{\sum_{j=1}^n K(\bar{\rho}(x, X_j)/h)} \cdot Y_i = \sum_{i=1}^n w_i(x) Y_i, \tag{1}$$

for some metric $\bar{\rho}$ and a bandwidth parameter $h$.

For the kernel regressor $f_{n,h}$ used to learn the metric $\rho$ below, $\bar{\rho}$ is the Euclidean metric. In the analysis we assume the bandwidth for $f_{n,h}$ is set as $h \geq \left( \log^2(n/\delta)/n \right)^{1/d}$, given a confidence

parameter $0 < \delta < 1$. In practice we would learn $h$ by cross-validation, but for the analysis we only need to know the existence of a good setting of $h$.

The metric is defined as

$$\mathbf{W}_i \triangleq \mathbb{E}_n \frac{|f_{n,h}(X + te_i) - f_{n,h}(X - te_i)|}{2t} \cdot \mathbf{1}_{\{A_{n,i}(X)\}} = \mathbb{E}_n \left[\Delta_{t,i} f_{n,h}(X) \cdot \mathbf{1}_{\{A_{n,i}(X)\}}\right], \quad (2)$$

where $A_{n,i}(X)$ is the event that *enough* samples contribute to the estimate $\Delta_{t,i} f_{n,h}(X)$. For the consistency result, we assume the following setting:

$$A_{n,i}(X) \equiv \min_{s \in \{-t,t\}} \mu_n(B(X + se_i, h/2)) \geq \alpha_n \text{ where } \alpha_n \triangleq \frac{2d \ln 2n + \ln(4/\delta)}{n}.$$

# 4 Consistency of the estimator $\mathbf{W}_i$ of $\|f_i'\|_{1,\mu}$

## 4.1 Theoretical setup

### 4.1.1 Marginal $\mu$

Without loss of generality we assume $\mathcal{X}$ has bounded diameter 1. The marginal is assumed to have a continuous density on $\mathcal{X}$ and has mass everywhere on $\mathcal{X}$: $\forall x \in \mathcal{X}, \forall h > 0, \mu(B(x,h)) \geq C_\mu h^d$. This is for instance the case if $\mu$ has a lower-bounded density on $\mathcal{X}$. Under this assumption, for samples $X$ in dense regions, $X \pm te_i$ is also likely to be in a dense region.

### 4.1.2 Regression function and noise

The output $Y \in \mathbb{R}$ is given as $Y = f(X) + \eta(X)$, where $\mathbb{E}\eta(X) = 0$. We assume the following general noise model: $\forall \delta > 0$ there exists $c > 0$ such that $\sup_{x \in \mathcal{X}} \mathbb{P}_{Y|X=x}(|\eta(x)| > c) \leq \delta$.

We denote by $C_Y(\delta)$ the infimum over all such $c$. For instance, suppose $\eta(X)$ has exponentially decreasing tail, then $\forall \delta > 0$, $C_Y(\delta) \leq O(\ln 1/\delta)$. A last assumption on the noise is that the variance of $(Y|X = x)$ is upper-bounded by a constant $\sigma_Y^2$ uniformly over all $x \in \mathcal{X}$.

Define the $\tau$-*envelope* of $\mathcal{X}$ as $\mathcal{X} + B(0,\tau) \triangleq \{z \in B(x,\tau), x \in \mathcal{X}\}$. We assume there exists $\tau$ such that $f$ is continuously differentiable on the $\tau$-envelope $\mathcal{X} + B(0,\tau)$. Furthermore, each derivative $f_i'(x) = e_i^\top \nabla f(x)$ is upper bounded on $\mathcal{X} + B(0,\tau)$ by $|f_i'|_{\sup}$ and is uniformly continuous on $\mathcal{X} + B(0,\tau)$ (this is automatically the case if the support $\mathcal{X}$ is compact).

### 4.1.3 Parameters varying with t

Our consistency results are expressed in terms of the following distributional quantities. For $i \in [d]$, define the $(t,i)$-*boundary* of $\mathcal{X}$ as $\partial_{t,i}(\mathcal{X}) \triangleq \{x : \{x + te_i, x - te_i\} \nsubseteq \mathcal{X}\}$. The smaller the mass $\mu(\partial_{t,i}(\mathcal{X}))$ at the boundary, the better we approximate $\|f_i'\|_{1,\mu}$.

The second type of quantity is $\epsilon_{t,i} \triangleq \sup_{x \in \mathcal{X}, s \in [-t,t]} |f_i'(x) - f_i'(x + se_i)|$.

Since $\mu$ has continuous density on $\mathcal{X}$ and $\nabla f$ is uniformly continuous on $\mathcal{X} + B(0,\tau)$, we automatically have $\mu(\partial_{t,i}(\mathcal{X})) \xrightarrow{t \to 0} 0$ and $\epsilon_{t,i} \xrightarrow{t \to 0} 0$.

## 4.2 Main theorem

Our main theorem bounds the error in estimating each norm $\|f_i'\|_{1,\mu}$ with $\mathbf{W}_i$. The main technical hurdles are in handling the various sample inter-dependencies introduced by both the estimates $f_{n,h}(X)$ and the events $A_{n,i}(X)$, and in analyzing the estimates at the boundary of $\mathcal{X}$.

**Theorem 1.** *Let $t + h \leq \tau$, and let $0 < \delta < 1$. There exist $C = C(\mu, K(\cdot))$ and $N = N(\mu)$ such that the following holds with probability at least $1 - 2\delta$. Define $A(n) \triangleq Cd \cdot \log(n/\delta) \cdot C_Y^2(\delta/2n) \cdot \sigma_Y^2 / \log^2(n/\delta)$. Let $n \geq N$, we have for all $i \in [d]$:*

$$\left|\mathbf{W}_i - \|f_i'\|_{1,\mu}\right| \leq \frac{1}{t}\left(\sqrt{\frac{A(n)}{nh^d}} + h \cdot \sum_{i \in [d]} |f_i'|_{sup}\right) + 2|f_i'|_{sup}\left(\sqrt{\frac{\ln 2d/\delta}{n}} + \mu(\partial_{t,i}(\mathcal{X}))\right) + \epsilon_{t,i}.$$

The bound suggest to set $t$ in the order of $h$ or larger. We need $t$ to be small in order for $\mu\left(\partial_{t,i}(\mathcal{X})\right)$ and $\epsilon_{t,i}$ to be small, but $t$ need to be sufficiently large (relative to $h$) for the estimates $f_{n,h}(X + te_i)$ and $f_{n,h}(X - te_i)$ to differ sufficiently so as to capture the variation in $f$ along $e_i$.

The theorem immediately implies consistency for $t \xrightarrow{n\to\infty} 0$, $h \xrightarrow{n\to\infty} 0$, $h/t \xrightarrow{n\to\infty} 0$, and $(n/\log n)h^d t^2 \xrightarrow{n\to\infty} \infty$. This is satisfied for many settings, for example $t \propto \sqrt{h}$ and $h \propto 1/\log n$.

### 4.3   Proof of Theorem 1

The main difficulty in bounding $\left|\mathbf{W}_i - \|f_i'\|_{1,\mu}\right|$ is in circumventing certain depencies: both quantities $f_{n,h}(X)$ and $A_{n,i}(X)$ depend not just on $X \in \mathbf{X}$, but on other samples in $\mathbf{X}$, and thus introduce inter-dependencies between the estimates $\Delta_{t,i} f_{n,h}(X)$ for different $X \in \mathbf{X}$.

To handle these dependencies, we carefully decompose $\left|\mathbf{W}_i - \|f_i'\|_{1,\mu}\right|$, $i \in [d]$, starting with:

$$\left|\mathbf{W}_i - \|f_i'\|_{1,\mu}\right| \le |\mathbf{W}_i - \mathbb{E}_n|f_i'(X)|| + \left|\mathbb{E}_n|f_i'(X)| - \|f_i'\|_{1,\mu}\right|. \tag{3}$$

The following simple lemma bounds the second term of (3).

**Lemma 3.** *With probability at least $1 - \delta$, we have for all $i \in [d]$,*

$$\left|\mathbb{E}_n|f_i'(X)| - \|f_i'\|_{1,\mu}\right| \le |f_i'|_{sup} \cdot \sqrt{\frac{\ln 2d/\delta}{n}}.$$

*Proof.* Apply a Chernoff bound, and a union bound on $i \in [d]$. $\qquad\square$

Now the first term of equation (3) can be further bounded as

$$|\mathbf{W}_i - \mathbb{E}_n|f_i'(X)|| \le \left|\mathbf{W}_i - \mathbb{E}_n|f_i'(X)| \cdot \mathbf{1}_{\{A_{n,i}(X)\}}\right| + \mathbb{E}_n|f_i'(X)| \cdot \mathbf{1}_{\{\bar{A}_{n,i}(X)\}}$$

$$\le \left|\mathbf{W}_i - \mathbb{E}_n|f_i'(X)| \cdot \mathbf{1}_{\{A_{n,i}(X)\}}\right| + |f_i'|_{\sup} \cdot \mathbb{E}_n\mathbf{1}_{\{\bar{A}_{n,i}(X)\}}. \tag{4}$$

We will bound each term of (4) separately.

The next lemma bounds the second term of (4). It is proved in the appendix. The main technicality in this lemma is that, for any $X$ in the sample $\mathbf{X}$, the event $\bar{A}_{n,i}(X)$ depends on other samples in $\mathbf{X}$.

**Lemma 4.** *Let $\partial_{t,i}(\mathcal{X})$ be defined as in Section (4.1.3). For $n \ge n(\mu)$, with probability at least $1 - 2\delta$, we have for all $i \in [d]$,*

$$\mathbb{E}_n\mathbf{1}_{\{\bar{A}_{n,i}(X)\}} \le \sqrt{\frac{\ln 2d/\delta}{n}} + \mu\left(\partial_{t,i}(\mathcal{X})\right).$$

It remains to bound $\left|\mathbf{W}_i - \mathbb{E}_n|f_i'(X)| \cdot \mathbf{1}_{\{A_{n,i}(X)\}}\right|$. To this end we need to bring in $f$ through the following quantities:

$$\widetilde{\mathbf{W}}_i \triangleq \mathbb{E}_n\left[\frac{|f(X + te_i) - f(X - te_i)|}{2t} \cdot \mathbf{1}_{\{A_{n,i}(X)\}}\right] = \mathbb{E}_n\left[\Delta_{t,i}f(X) \cdot \mathbf{1}_{\{A_{n,i}(X)\}}\right]$$

and for any $x \in \mathcal{X}$, define $\tilde{f}_{n,h}(x) \triangleq \mathbb{E}_{\mathbf{Y}|\mathbf{X}}f_{n,h}(x) = \sum_i w_i(x)f(x_i)$.

The quantity $\widetilde{\mathbf{W}}_i$ is easily related to $\mathbb{E}_n|f_i'(X)| \cdot \mathbf{1}_{\{A_{n,i}(X)\}}$. This is done in Lemma 5 below. The quantity $\tilde{f}_{n,h}(x)$ is needed when relating $\mathbf{W}_i$ to $\widetilde{\mathbf{W}}_i$.

**Lemma 5.** *Define $\epsilon_{t,i}$ as in Section (4.1.3). With probability at least $1 - \delta$, we have for all $i \in [d]$,*

$$\left|\widetilde{\mathbf{W}}_i - \mathbb{E}_n|f_i'(X)| \cdot \mathbf{1}_{\{A_{n,i}(X)\}}\right| \le \epsilon_{t,i}.$$

*Proof.* We have $f(x + te_i) - f(x - tei) = \int_{-t}^{t} f'_i(x + se_i)\, ds$ and therefore

$$2t\left(f'_i(x) - \epsilon_{t,i}\right) \le f(x + te_i) - f(x - tei) \le 2t\left(f'_i(x) + \epsilon_{t,i}\right).$$

It follows that $\left|\frac{1}{2t}\left|f(x + te_i) - f(x - tei)\right| - |f'_i(x)|\right| \le \epsilon_{t,i}$, therefore

$$\left|\widetilde{\mathbf{W}}_i - \mathbb{E}_n\,|f'_i(X)| \cdot \mathbf{1}_{\{A_{n,i}(X)\}}\right| \le \mathbb{E}_n\left|\frac{1}{2t}\left|f(x + te_i) - f(x - tei)\right| - |f'_i(x)|\right| \le \epsilon_{t,i}.$$

$\square$

It remains to relate $\mathbf{W}_i$ to $\widetilde{\mathbf{W}}_i$. We have

$$
\begin{aligned}
2t\left|\mathbf{W}_i - \widetilde{\mathbf{W}}_i\right| &= 2t\left|\mathbb{E}_n(\Delta_{t,i}f_{n,h}(X) - \Delta_{t,i}f(X)) \cdot \mathbf{1}_{\{A_{n,i}(X)\}}\right| \\
&\le 2\max_{s \in \{-t,t\}} \mathbb{E}_n|f_{n,h}(X + se_i) - f(X + se_i)| \cdot \mathbf{1}_{\{A_{n,i}(X)\}} \\
&\le 2\max_{s \in \{-t,t\}} \mathbb{E}_n\left|f_{n,h}(X + se_i) - \tilde{f}_{n,h}(X + se_i)\right| \cdot \mathbf{1}_{\{A_{n,i}(X)\}} && (5)\\
&\quad + 2\max_{s \in \{-t,t\}} \mathbb{E}_n\left|\tilde{f}_{n,h}(X + se_i) - f(X + se_i)\right| \cdot \mathbf{1}_{\{A_{n,i}(X)\}}. && (6)
\end{aligned}
$$

We first handle the bias term (6) in the next lemma which is given in the appendix.

**Lemma 6** (Bias). *Let $t + h \le \tau$. We have for all $i \in [d]$, and all $s \in \{t, -t\}$:*

$$\mathbb{E}_n\left|\tilde{f}_{n,h}(X + se_i) - f(X + se_i)\right| \cdot \mathbf{1}_{\{A_{n,i}(X)\}} \le h \cdot \sum_{i \in [d]} |f'_i|_{sup}.$$

The variance term in (5) is handled in the lemma below. The proof is given in the appendix.

**Lemma 7** (Variance terms). *There exist $C = C(\mu, K(\cdot))$ such that, with probability at least $1 - 2\delta$, we have for all $i \in [d]$, and all $s \in \{-t, t\}$:*

$$\mathbb{E}_n\left|f_{n,h}(X + se_i) - \tilde{f}_{n,h}(X + se_i)\right| \cdot \mathbf{1}_{\{A_{n,i}(X)\}} \le \sqrt{\frac{Cd \cdot \log(n/\delta)C_Y^2(\delta/2n) \cdot \sigma_Y^2}{n(h/2)^d}}.$$

The next lemma summarizes the above results:

**Lemma 8.** *Let $t + h \le \tau$ and let $0 < \delta < 1$. There exist $C = C(\mu, K(\cdot))$ such that the following holds with probability at least $1 - 2\delta$. Define $A(n) \triangleq Cd \cdot \log(n/\delta) \cdot C_Y^2(\delta/2n) \cdot \sigma_Y^2 / \log^2(n/\delta)$. We have*

$$\left|\mathbf{W}_i - \mathbb{E}_n\,|f'_i(X)| \cdot \mathbf{1}_{\{A_{n,i}(X)\}}\right| \le \frac{1}{t}\left(\sqrt{\frac{A(n)}{nh^d}} + h \cdot \sum_{i \in [d]} |f'_i|_{sup}\right) + \epsilon_{t,i}.$$

*Proof.* Apply lemmas 5, 6 and 7, in combination with equations 5 and 6. $\square$

To complete the proof of Theorem 1, apply lemmas 8 and 3 in combination with equations 3 and 4.

## 5 Experiments

### 5.1 Data description

We present experiments on several real-world regression datasets. The first two datasets describe the dynamics of 7 degrees of freedom of robotic arms, Barrett WAM and SARCOS [9, 10]. The input points are 21-dimensional and correspond to samples of the positions, velocities, and accelerations of the 7 joints. The output points correspond to the torque of each joint. The far joints (1, 5, 7)

|  | Barrett joint 1 | Barrett joint 5 | SARCOS joint 1 | SARCOS joint 5 | Housing |
|---|---|---|---|---|---|
| KR error | $0.50 \pm 0.02$ | $0.50 \pm 0.03$ | $0.16 \pm 0.02$ | $0.14 \pm 0.02$ | $0.37 \pm 0.08$ |
| KR-$\rho$ error | $\mathbf{0.38} \pm 0.03$ | $\mathbf{0.35} \pm 0.02$ | $\mathbf{0.14} \pm 0.02$ | $\mathbf{0.12} \pm 0.01$ | $\mathbf{0.25} \pm 0.06$ |
| KR time | $0.39 \pm 0.02$ | $0.37 \pm 0.01$ | $0.28 \pm 0.05$ | $0.23 \pm 0.03$ | $0.10 \pm 0.01$ |
| KR-$\rho$ time | $0.41 \pm 0.03$ | $0.38 \pm 0.02$ | $0.32 \pm 0.05$ | $0.23 \pm 0.02$ | $0.11 \pm 0.01$ |
|  | Concrete Strength | Wine Quality | Telecom | Ailerons | Parkinson's |
| KR error | $0.42 \pm 0.05$ | $\mathbf{0.75} \pm 0.03$ | $0.30 \pm 0.02$ | $0.40 \pm 0.02$ | $0.38 \pm 0.03$ |
| KR-$\rho$ error | $\mathbf{0.37} \pm 0.03$ | $\mathbf{0.75} \pm 0.02$ | $\mathbf{0.23} \pm 0.02$ | $\mathbf{0.39} \pm 0.02$ | $\mathbf{0.34} \pm 0.03$ |
| KR time | $0.14 \pm 0.02$ | $0.19 \pm 0.02$ | $0.15 \pm 0.01$ | $0.20 \pm 0.01$ | $0.30 \pm 0.03$ |
| KR-$\rho$ time | $0.14 \pm 0.01$ | $0.19 \pm 0.02$ | $0.16 \pm 0.01$ | $0.21 \pm 0.01$ | $0.30 \pm 0.03$ |

|  | Barrett joint 1 | Barrett joint 5 | SARCOS joint 1 | SARCOS joint 5 | Housing |
|---|---|---|---|---|---|
| $k$-NN error | $0.41 \pm 0.02$ | $0.40 \pm 0.02$ | $0.08 \pm 0.01$ | $0.08 \pm 0.01$ | $0.28 \pm 0.09$ |
| $k$-NN-$\rho$ error | $\mathbf{0.29} \pm 0.01$ | $\mathbf{0.30} \pm 0.02$ | $\mathbf{0.07} \pm 0.01$ | $\mathbf{0.07} \pm 0.01$ | $\mathbf{0.22} \pm 0.06$ |
| $k$-NN time | $0.21 \pm 0.04$ | $0.16 \pm 0.03$ | $0.13 \pm 0.01$ | $0.13 \pm 0.01$ | $0.08 \pm 0.01$ |
| $k$-NN-$\rho$ time | $0.13 \pm 0.04$ | $0.16 \pm 0.03$ | $0.14 \pm 0.01$ | $0.13 \pm 0.01$ | $0.08 \pm 0.01$ |
|  | Concrete Strength | Wine Quality | Telecom | Ailerons | Parkinson's |
| $k$-NN error | $0.40 \pm 0.04$ | $0.73 \pm 0.04$ | $\mathbf{0.13} \pm 0.02$ | $0.37 \pm 0.01$ | $0.22 \pm 0.01$ |
| $k$-NN-$\rho$ error | $\mathbf{0.38} \pm 0.03$ | $\mathbf{0.72} \pm 0.03$ | $0.17 \pm 0.02$ | $\mathbf{0.34} \pm 0.01$ | $\mathbf{0.20} \pm 0.01$ |
| $k$-NN time | $0.10 \pm 0.01$ | $0.15 \pm 0.01$ | $0.16 \pm 0.02$ | $0.12 \pm 0.01$ | $0.14 \pm 0.01$ |
| $k$-NN-$\rho$ time | $0.11 \pm 0.01$ | $0.15 \pm 0.01$ | $0.15 \pm 0.01$ | $0.11 \pm 0.01$ | $0.15 \pm 0.01$ |

Table 1: Normalized mean square prediction errors and average prediction time per point (in milliseconds). The top two tables are for KR vs KR-$\rho$ and the bottom two for $k$-NN vs $k$-NN-$\rho$.

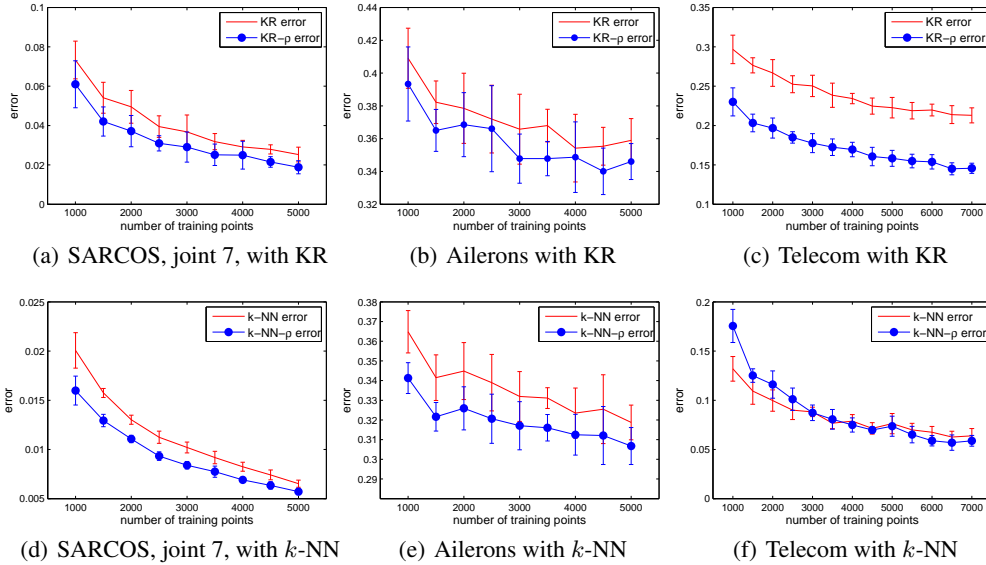

(a) SARCOS, joint 7, with KR     (b) Ailerons with KR     (c) Telecom with KR

(d) SARCOS, joint 7, with $k$-NN     (e) Ailerons with $k$-NN     (f) Telecom with $k$-NN

Figure 2: Normalized mean square prediction error over 2000 points for varying training sizes. Results are shown for $k$-NN and kernel regression (KR), with and without the metric $\rho$.

correspond to different regression problems and are the only results reported. Expectedly, results for the other joints are similarly good.

The other datasets are taken from the UCI repository [11] and from [12]. The concrete strength dataset (Concrete Strength) contains 8-dimensional input points, describing age and ingredients of concrete, the output points are the compressive strength. The wine quality dataset (Wine Quality) contains 11-dimensional input points corresponding to the physicochemistry of wine samples, the output points are the wine quality. The ailerons dataset (Ailerons) is taken from the problem of flying a F16 aircraft. The 5-dimensional input points describe the status of the aeroplane, while the goal is

to predict the control action on the ailerons of the aircraft. The housing dataset (Housing) concerns the task of predicting housing values in areas of Boston, the input points are 13-dimensional. The Parkinson's Telemonitoring dataset (Parkison's) is used to predict the clinician's Parkinson's disease symptom score using biomedical voice measurements represented by 21-dimensional input points. We also consider a telecommunication problem (Telecom), wherein the 47-dimensional input points and the output points describe the bandwidth usage in a network.

For all datasets we normalize each coordinate with its standard deviation from the training data.

## 5.2 Experimental setup

To learn the metric, we set $h$ by cross-validation on half the training points, and we set $t = h/2$ for all datasets. Note that in practice we might want to also tune $t$ in the range of $h$ for even better performance than reported here. The event $A_{n,i}(X)$ is set to reject the gradient estimate $\Delta_{n,i} f_{n,h}(X)$ at $X$ if no sample contributed to one the estimates $f_{n,h}(X \pm te_i)$.

In each experiment, we compare kernel regression in the euclidean metric space (KR) and in the learned metric space (KR-$\rho$), where we use a box kernel for both. Similar comparisons are made using $k$-NN and $k$-NN-$\rho$. All methods are implemented using a fast neighborhood search procedure, namely the cover-tree of [13], and we also report the average prediction times so as to confirm that, on average, time-performance is not affected by using the metric.

The parameter $k$ in $k$-NN/$k$-NN-$\rho$, and the bandwidth in KR/KR-$\rho$ are learned by cross-validation on half of the training points. We try the same range of $k$ (from 1 to $5 \log n$) for both $k$-NN and $k$-NN-$\rho$. We try the same range of bandwidth/space-diameter (a grid of size 0.02 from 1 to 0.02 ) for both KR and KR-$\rho$: this is done efficiently by starting with a log search to detect a smaller range, followed by a grid search on a smaller range.

Table 5 shows the normalized Mean Square Errors (nMSE) where the MSE on the test set is normalized by variance of the test output. We use 1000 training points in the robotic datasets, 2000 training points in the Telecom, Parkinson's, Wine Quality, and Ailerons datasets, and 730 training points in Concrete Strength, and 300 in Housing. We used 2000 test points in all of the problems, except for Concrete, 300 points, and Housing, 200 points. Averages over 10 random experiments are reported.

For the larger datasets (SARCOS, Ailerons, Telecom) we also report the behavior of the algorithms, with and without metric, as the training size $n$ increases (Figure 2).

## 5.3 Discussion of results

From the results in Table 5 we see that virtually on all datasets the metric helps improve the performance of the distance based-regressor even though we did not tune $t$ to the particular problem (remember $t = h/2$ for all experiments). The only exceptions are for Wine Quality where the learned weights are nearly uniform, and for Telecom with $k$-NN. We noticed that the Telecom dataset has a lot of outliers and this probably explains the discrepancy, besides from the fact that we did not attempt to tune $t$. Also notice that the error of $k$-NN is already low for small sample sizes, making it harder to outperform. However, as shown in Figure 2, for larger training sizes $k$-NN-$\rho$ gains on $k$-NN. The rest of the results in Figure 2 where we vary $n$ are self-descriptive: gradient weighting clearly improves the performance of the distance-based regressors.

We also report the average prediction times in Table 5. We see that running the distance-based methods with gradient weights does not affect estimation time. Last, remember that the metric can be learned online at the cost of only $2d$ times the average kernel estimation time reported.

## 6 Final remarks

Gradient weighting is simple to implement, computationally efficient in batch-mode and online, and most importantly improves the performance of distance-based regressors on real-world applications. In our experiments, most or all coordinates of the data are relevant, yet some coordinates are more important than others. This is sufficient for gradient weighting to yield gains in performance. We believe there is yet room for improvement given the simplicity of our current method.

## Footnotes

*Currently at Toyota Technological Institute Chicago, and affiliated with the Max Planck Institute.

[1] Accounting for the scale change induced by $\rho$ on the space $\mathcal{X}$.

# References

[1] Kilian Q. Weinberger and Gerald Tesauro. Metric learning for kernel regression. *Journal of Machine Learning Research - Proceedings Track*, 2:612–619, 2007.

[2] Bo Xiao, Xiaokang Yang, Yi Xu, and Hongyuan Zha. Learning distance metric for regression by semidefinite programming with application to human age estimation. In *Proceedings of the 17th ACM international conference on Multimedia*, pages 451–460, 2009.

[3] Shai Shalev-shwartz, Yoram Singer, and Andrew Y. Ng. Online and batch learning of pseudo-metrics. In *ICML*, pages 743–750. ACM Press, 2004.

[4] Jason V. Davis, Brian Kulis, Prateek Jain, Suvrit Sra, and Inderjit S. Dhillon. Information-theoretic metric learning. In *ICML*, pages 209–216, 2007.

[5] W. Härdle and T. Gasser. On robust kernel estimation of derivatives of regression functions. *Scandinavian journal of statistics*, pages 233–240, 1985.

[6] J. Lafferty and L. Wasserman. Rodeo: Sparse nonparametric regression in high dimensions. *Arxiv preprint math/0506342*, 2005.

[7] L. Rosasco, S. Villa, S. Mosci, M. Santoro, and A. Verri. Nonparametric sparsity and regularization. *http://arxiv.org/abs/1208.2572*, 2012.

[8] L. Gyorfi, M. Kohler, A. Krzyzak, and H. Walk. *A Distribution Free Theory of Nonparametric Regression*. Springer, New York, NY, 2002.

[9] S. Kpotufe. k-NN Regression Adapts to Local Intrinsic Dimension. *NIPS*, 2011.

[10] Duy Nguyen-Tuong, Matthias W. Seeger, and Jan Peters. Model learning with local gaussian process regression. *Advanced Robotics*, 23(15):2015–2034, 2009.

[11] Duy Nguyen-Tuong and Jan Peters. Incremental online sparsification for model learning in real-time robot control. *Neurocomputing*, 74(11):1859–1867, 2011.

[12] A. Frank and A. Asuncion. UCI machine learning repository. `http://archive.ics.uci.edu/ml`. University of California, Irvine, School of Information and Computer Sciences, 2012.

[13] Luis Torgo. Regression datasets. `http://www.liaad.up.pt/~ltorgo`. University of Porto, Department of Computer Science, 2012.

[14] A. Beygelzimer, S. Kakade, and J. Langford. Cover trees for nearest neighbors. *ICML*, 2006.

